# EfficientCAPER: An End-to-End Framework for Fast and Robust Category-Level Articulated Object Pose Estimation

**Xinyi Yu[1], Haonan Jiang[1], Li Zhang[2], Lin Yuanbo Wu[3], Linin Ou[1]\*, Liu Liu[4]\***

1 Zhejiang University of Technology, Zhejiang, China
2 University of Science and Technology of China, Hefei, China
3 Swansea University, Swansea, UK
4 Hefei University of Technology, Hefei, China
211122030127@zjut.edu.cn, liuliu@hfut.edu.cn

## Abstract

Human life is populated with articulated objects. Pose estimation for category-level articulated objects is a significant challenge due to their inherent complexity and diverse kinematic structures. Current methods for this task usually meet the problems of insufficient consideration of kinematic constraints, self-occlusion, and optimization requirements. In this paper, we propose **EfficientCAPER**, an end-to-end **C**ategory-level **A**rticulated object **P**ose **E**stimato**R**, eliminating the need for optimization functions as post-processing and utilizing the kinematic structure for joint-centric pose modeling, thus enhancing the efficiency and applicability. Given a partial point cloud as input, the EfficientCAPER firstly estimates the pose for the free part of an articulated object using decoupled rotation representation. Next, we canonicalize the input point cloud to estimate constrained parts' poses by predicting the joint parameters and states as replacements. Evaluations on three diverse datasets, ArtImage, ReArtMix, and RobotArm, show EfficientCAPER's effectiveness and generalization ability to real-world scenarios. The framework exhibits excellent static pose estimation performance for articulated objects, contributing to the advancement of category-level pose estimation. Code is available at https://github.com/wxycwymds/EfficientCAPER.

## 1 Introduction

Articulated objects are everywhere in our daily lives, from small items like eyeglasses to larger entities like dishwashers. Unlike rigid objects, which are treated as single entities in three-dimensional space, articulated objects are made up of several movable rigid parts connected by joints, following a specific kinematic structure. Estimating the 6D pose—meaning the 3D rotation and translation for each rigid part from visual observation of an articulated object is a fundamental challenge in computer vision. This task is crucial for various applications in robotics [1], augmented reality [2], virtual reality [3], and 3D scene understanding [4].

Generally, compared to instance-level 6D pose estimation [5, 6, 7], category-level pose estimation requires machines to understand the 3D rotation, 3D translation, and 3D scale of unseen objects, using only semantic category prior.

In recent years, there has been a growing interest in the task of category-level articulated object pose estimation [8, 9, 10, 11]. Representative methods such as Normalized Object Coordinate Space (NOCS) and its variants [12, 13] have emerged as leading approaches. These methods introduce

intra-class or intra-part representations for each rigid part in canonical space and recover the object poses by estimating these representations. This approach has become the mainstream paradigm for investigating category-level articulated objects. However, this solution faces several challenges:

(1) Most of them adopt a part-centric approach for pose modeling, where each part's pose is estimated independently, disregarding the constraints imposed by the kinematic structure. (2) They often struggle with self-occlusion cases, especially where smaller parts are obscured by larger parts from certain camera perspectives. (3) The introduced intra-class or intra-part representations are often intermediate, requiring additional optimization to obtain the final poses, which can be time-consuming.

In fact, all the rigid parts of an articulated object can be categorized into two groups: free parts, which can move into any rotational and translational target (e.g., the main body of a cabinet) without kinematic constraints (with an articulated object containing only one free part), and constrained parts, which can rotate or translate along a pre-defined joint (e.g., cabinet doors or drawers) following physically reasonable kinematic laws (with any number of constrained parts in an articulated object). Building on this insight, we propose a joint-centric articulation pose modeling mechanism that establishes part-joint pairs. This approach allows learning each part's pose as a joint state representation, which is easier for a neural network to learn compared to direct 6D pose regression or intra-class/intra-part representation. Furthermore, this joint state representation significantly improves visually occluded part pose estimation, as even a few visible points can robustly contribute to joint state prediction.

In this paper, within the joint-centric pose modeling mechanism, we propose **EfficientCAPER**, a novel end-to-end **Efficient C**ategory-level **A**rticulated object **P**ose **E**stimato**R**. EfficientCAPER breaks down pose estimation into two stages: estimating the pose of the free part and the constrained parts, without the need for intermediate variable estimation or optimization procedures. This departure from traditional approaches, such as A-NCSH [8] and OMAD [14], eliminates the reliance on intermediate steps that recover pose using optimistic equations or retrieval methods.

Specifically, in the first stage, we utilize a decomposition strategy to represent the rotation matrix, as described in [15]. This strategy enables us to estimate confidence-aware rotation vectors using a hybrid feature extraction encoder (HS-encoder [16]), which effectively extracts essential geometric features and relationships. In the second stage, we predict joint states, joint parameters, and the 3D scale of each part instead of using intermediate representations. By integrating these stages, EfficientCAPER achieves comprehensive category-level pose estimation for articulated objects, providing a streamlined and efficient approach. We evaluate our EfficientCAPER on three datasets including the synthetic dataset ArtImage [14], and the semi-synthetic dataset ReArtMix [13]. We also infer the model on a RobotArm dataset [13] that contains much more diverse and complex scenes, which demonstrates that EfficientCAPER has the generalization ability to real-world scenarios. Through these extensive experiments, we provide evidence of the EfficientCAPER's superior performance in static pose estimation of articulated objects. In summary, our main contributions are as follows:

- We proposed EfficientCAPER, an end-to-end framework for estimating category-level articulated object 6D pose efficiently and robustly without any optimization or other post-processing.

- Our EfficientCAPER solves articulation pose estimation by two stages that firstly estimates the pose of the free part with decoupled rotation representation and secondly predicts joint states as pose replacement. We name this process as joint-centric pose modeling.

- The efficiency and robustness of the EfficientCAPER are demonstrated through the evaluation of either synthetic or real-world articulated object benchmarks. The sufficient experiments show the dramatic performance improvement and generalization capacity of our method.

## 2 Related Work

### 2.1 Rigid Objects Pose Estimation

Rigid object pose estimation has been a significant research area in computer vision, with numerous approaches proposed to tackle this challenging problem. Traditional methods often rely on template-

based methods [17, 18] or feature-based techniques such as SIFT [19] and SURF [20]. However, these methods are sensitive to occlusions and variations in illumination. Recent advancements in deep learning have revolutionized rigid object pose estimation. Techniques employing convolutional neural networks (CNNs) have shown remarkable performance improvements. For instance, [21] proposed a method based on a CNN architecture that directly regresses 6-DoF poses from RGB images. While considerable progress has been made in rigid object pose estimation, the extension to category-level pose estimation remains relatively unexplored. Category-level pose estimation aims to estimate the pose of objects belonging to specific categories, despite the intra-category variations. NOCS [12] is the first method for category-level pose estimation by matching per-pixel correspondences between the observed point cloud and the estimated normalized canonical coordinates. [22] utilize variational auto-encoder to capture both pose-independent and pose-dependent features to directly predict the 6D pose. Given a single scene image, SAR-Net [23] proposed a method that relies on the shape information from the depth (D) channel, without using external real pose-annotated training data. GPV-Pose [24] presents geometric-pose consistency terms and point-wise bounding box voting with backbone 3D-GC [25]. Based on GPV-Pose, HS-Pose [16] proposed a network structure named HS-layer that extends 3D-GC to extract hybrid scope latent features from point cloud data. Due to HS-layer's ability to perceive both local and global geometric information and its robustness to noise, we have chosen the HS-Encoder as our backbone.

## 2.2 Category-level Articulation Pose Estimation

Articulation pose estimation has been a topic of significant interest in computer vision and robotics. In addition to the definition of rigid object pose estimation, articulated objects consist of a finite number of rigid parts that are interconnected through various types of joints. For articulation pose estimation, we need per-part 6D pose as estimated results. Recent advancements in deep learning have led to significant progress in articulation pose estimation. Convolutional neural networks (CNNs) have been particularly successful in learning complex spatial relationships and patterns from raw data. For instance, [26] proposed a method based on a multi-stage CNN architecture for human pose estimation, achieving state-of-the-art accuracy on benchmark datasets such as MPII Human Pose and COCO. Similarly, [27] introduced a pose estimation framework using graph convolutional networks (GCNs) to model the spatial dependencies between body joints, demonstrating robust performance under occlusions and challenging poses. While existing research has focused on articulation pose estimation for a specific instance and lacks practicality on articulation.

Category-level articulation pose estimation aims at predicting the pose of previously unseen articulated objects. Li et al. first propose A-NCSH [8], which extends the notation of normalized coordinates [12] into articulation. [13] introduce the concept of part pairs to investigate previously unseen instances by extending the setting of A-NCSH to real-world scenarios while studying articulated objects. In addition, Xue et al. first [14] presents utilizing key points as a method for articulation modeling, aiming to enhance the speed of inference while maintaining accurate pose estimation. Recently, [11] proposed a novel category-level pose estimation framework based on reinforcement learning. However, these methods are limited by the accuracy of the kinematic model and may struggle with real-world variability and occlusions.

## 3 Problem Statement

To achieve an end-to-end category-level articulated object pose estimation framework, our core idea is to introduce a two-stage end-to-end pose estimator EfficientCAPER without any post-processing or optimization operation. Given a 3D observed object point cloud $\mathcal{P} \in \mathbb{R}^3$ with $K$ rigid parts, our EfficientCAPER network performs predictions under unknown CAD models for (1) per-point part segmentation $\delta$, (2) 3D rotation $R_{\textbf{free}}$ and 3D translation $t_{\textbf{free}}$ for free part, in where the 3D rotation is encoded as a decoupled rotation representation (3) 3D rotation $R_{\textbf{cons}}^{(k)}$ and per-part 3D translation $t_{\textbf{cons}}^{(k)}$ for each constrained part, which are represented as joint state $\theta_{\textbf{cons}}^{(k)}$. (4) 3D scale for free part $s_{\textbf{free}}$ and constrained parts $s_{\textbf{cons}}^{(k)}$. In total, the rotation, translation and scale together constitute an articulation pose estimation result $\{R^{(k)}, t^{(k)}, s^{(k)}\}_{k=1}^{K}$, where $\{R^{(k)}\}_{k=1}^{K} = \{R_{\textbf{free}}, \{R_{\textbf{cons}}^{(k)}\}_{k=2}^{K}\}$, $\{t^{(k)}\}_{k=1}^{K} = \{t_{\textbf{free}}, \{t_{\textbf{cons}}^{(k)}\}_{k=2}^{K}\}$ and $\{s^{(k)}\}_{k=1}^{K} = \{s_{\textbf{free}}, \{s_{\textbf{cons}}^{(k)}\}_{k=2}^{K}\}$, where we assume that there is only one free part in an articulated object and the corresponding index is 1.

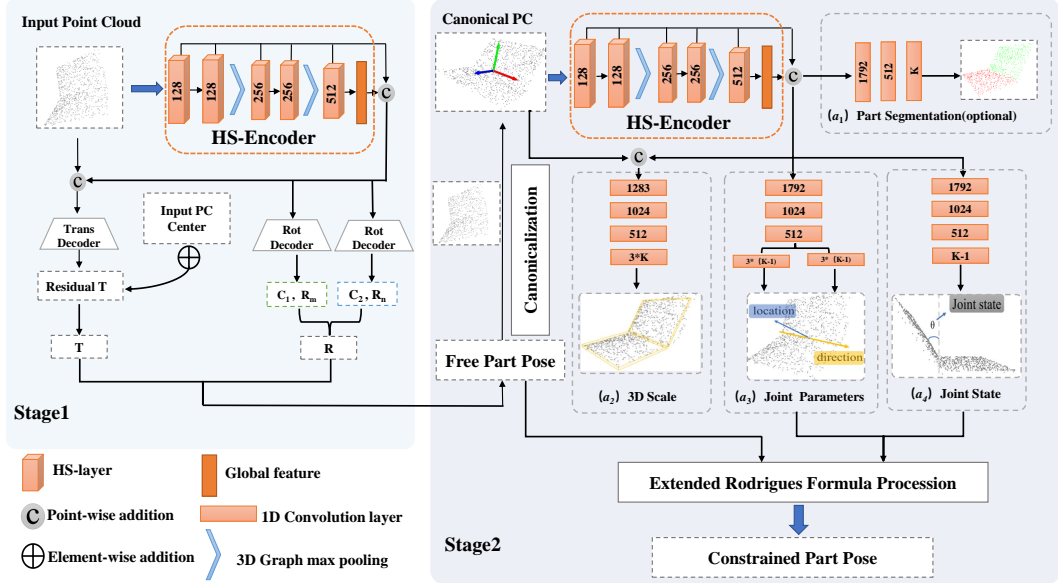

Figure 1: The EfficientCAPER architecture. Taking the partial point cloud of an articulated object as input, our network estimates the pose of the free part using HS-Encoder as the backbone and decoupled rotation as representation. In the second stage, we canonicalize the input point cloud and predict per-part scale, segmentation, joint parameters, and joint states, where the latter two are used to recover the constrained parts' poses.

# 4  Method

## 4.1  Overview

We present the overall pipeline of EfficientCAPER in Figure 1. Given the point cloud of an articulated object, our approach estimates the pose of its parts through two stages. First, we estimate the pose of a free part, and then we recover the poses of constrained parts based on joint state estimation and kinematic constraints. In the initial stage, we extract hybrid-scope features from the input point cloud using an HS-Encoder [16]. These features, along with the point cloud, are inputted into one trans-decoder and two rot-decoders to estimate the free part's translation and confidence-aware rotation vectors. Moving to the second stage, we canonicalize the input point cloud using the estimated pose of the free part, which improves the robustness of the estimation process. We then use another HS-Encoder to extract hybrid-scope features, which are fed into four subbranches for joint stage estimation, joint parameters estimation, scale estimation, and part segmentation respectively. Finally, the poses of the constrained parts are calculated based on the joint state and the free part's pose. Note that the input point cloud can be obtained from an RGB image and a depth image through back-projection.

## 4.2  Joint-Centric Articulation Pose Modeling

As discussed above, most of the articulation pose estimation methods adopt a part-centric pose modeling strategy that might ignore the effect of kinematic structure and also suffer from the self-occlusion problem. In this paper, we take a joint-centric perspective to investigate articulated objects, which regards the pose estimation task as a joint state prediction task. In this way, the motion of each constrained part corresponds to the joint state changing.

We consider two groups of rigid parts in an articulated object: the free part and the constrained part, where the former can move arbitrarily in three-dimensional space and the latter can move along the corresponding joint constrained by the kinematic structure. The constrained part's motion is determined by the joint type. Following the articulation setting in A-NCSH [8] and OMAD [14], we consider two types of joints: (1) Revolute joint that allows the rotational motion in the articulated object. The joint state $\theta_r$ is denoted by the relative rotation angle compared to the rest state, and the joint parameters can be defined as $\phi_r = (u_r, q_r)$, where $u_r$ represents the direction of the joint axis and $q_r$ represents the pivot point position of joint axis in the canonical coordinate space. (2)

Prismatic joint that allows the constrained part to move along the joint direction. The joint state $\boldsymbol{\theta}_p$ is denoted by the relative distance compared to the rest state, and the joint parameters can be defined as $\boldsymbol{\phi}_p = (\boldsymbol{u}_p)$, where $\boldsymbol{u}_p$ represents the direction of the joint axis. By defining the joint-centric pose modeling method, we can correspond part to joint one by one, and the total articulated object poses for all the $K$ parts can be represented by a sequence of joint states $\boldsymbol{\Theta} = \{\boldsymbol{\theta}^{(k)}\}_{k=2}^{K}$ of constrained parts and the pose $\{R_{\mathbf{free}}, \boldsymbol{t}_{\mathbf{free}}\}$ of free part. The definitions of free part and constrained part are illustrated in Figure 2.

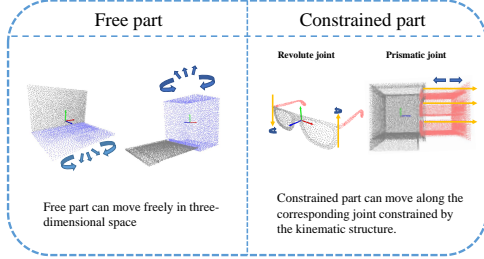

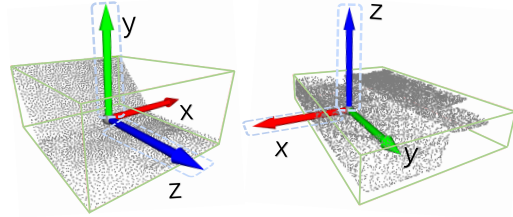

Figure 2: Free part and Constrained part in articulated objects. These two kinds of parts are connected by joints.

Figure 3: Rotation decoupled strategies. We select the corresponding two axes according to the symmetry priors.

### 4.3 Free Part Pose Estimation with Decoupled Rotation

In our EfficientCAPER, the input of the network is a partial point cloud of an articulated object $\mathcal{P} \in \mathbb{R}^{N \times 3}$. In the first stage, we aim to estimate the pose $\{R_{\mathbf{free}}, \boldsymbol{t}_{\mathbf{free}}\}$ for free part, and here we describe the rotation $R_{\mathbf{free}}$ and translation $\boldsymbol{t}_{\mathbf{free}}$ estimation separately.

**Rotation Estimation.** The GPV-Pose [24] was proposed to estimate the 6D pose with the rigid object point cloud input. When transferring into the free part pose estimation, we input the point cloud from all the rigid parts rather than only the free part's point cloud, since (1) the part segmentation is not required as a prerequisite task that further improves the efficiency of our method. (2) other constrained parts can provide auxiliary information for estimating the pose of the free part with strong symmetry. Specifically, given the point cloud $\mathcal{P} \in \mathbb{R}^{N \times 3}$, we build an encoder-decoder architecture which includes an HS-Encoder with four HS-layers and rot decoder with four 1-D convolution layers. We can extract the feature of each HS-layer after encoding, and we performed a max pooling operation on the point cloud dimension of the last layer's output to get a global feature that contains global information. We concatenate the global feature with the output of each HS layer as the ultimate output of the encoder. The $R_{\mathbf{free}}$ can be decomposed into three direction vectors: x, y, and z, we select two of them for different objects to help the network learn shape prior better. As shown in Figure 3, we choose y-axis and z-axis direction vectors since the laptop is symmetric about the $yOz$ plane. Similarly, we choose the x-axis and z-axis direction for the box. Inspired by [24], an attempt is made to estimate a confidence value for each vector to enhance the robustness during the recovery of the final 3D rotation. For the decoding process, we decode the ultimate feature to get two vectors, then we normalize them to get predicted plane normals and implement a softmax operation to get confidence for each predicted normal.

**Translation Estimation.** Regarding the translation $\boldsymbol{t}$, we firstly zero-center the input point cloud $\mathcal{P}$ by reducing the mean of point cloud $\widetilde{\mathcal{P}}$. Next, we build a similar decoder consisting of 1-D convolution layers. Inspired by [15], we estimate the residual between the mean value of $\mathcal{P}$ and the ground truth of translation. The translation feature is formed by concatenating the output of every HS-layer and zero-center point cloud, and we can finally get residual translation $\boldsymbol{t}_{\mathbf{free}}^{*}$ through the translation decoder. The final translation of free part $\boldsymbol{t}_{\mathbf{free}}$ can be obtained by $\boldsymbol{t}_{\mathbf{free}} = \boldsymbol{t}_{\mathbf{free}}^{*} + \widetilde{\mathcal{P}}$. Finally, the 6D pose of the free part is the constitution of rotation and translation $T_{\mathbf{free}} = \{R_{\mathbf{free}}, \boldsymbol{t}_{\mathbf{free}}\}$.

## 4.4   Articulation Pose Canonicalization

Before the second stage of estimating poses for constrained parts, we canonicalize the input point cloud with the predicted free part's pose $T_{\mathbf{free}}$. This operation provides the following benefits: (1) the joint state estimation task in camera space can be transformed into a "pseudo-canonical" space, which is more suitable for neural network learning. (2) the canonicalization process applied to the point cloud can effectively eliminate the effects of different joint configurations in the movable rigid parts, it allows the canonicalized point cloud to provide both a shape prior and a kinematic prior that make a significant contribution to regress joint states.

In articulated pose canonicalization module, the canonicalized point cloud $\hat{\mathcal{P}}$ is computed as the product of the inverse transformation $T_{\mathbf{free}}$ and the point cloud $P^{(k)}$:

$$\hat{\mathcal{P}}^{(k)} = (T_{\mathbf{free}})^{-1}\mathcal{P}^{(k)} = (R_{\mathbf{free}})^{-1}(\mathcal{P}^{(k)} - \boldsymbol{t}_{\mathbf{free}}) \tag{1}$$

By canonicalizing the input point cloud, the regressing model in stage two will take the canonicalized point cloud as input to estimate joint states, the neural network would increase sensitivity to parameters and states of joint and could considerably enhance regression performance.

## 4.5   Constrained Part Pose Estimation with Joint State Replacement

At the second stage of our EfficientCAPER, we predict the pose for constrained parts $\{R_{\mathbf{cons}}^{(k)}\}_{k=2}^{K}$ as well as scales $\{\boldsymbol{s}_{\mathbf{cons}}^{(k)}\}_{k=2}^{K}$, labels $\delta$ and joint parameters $(\boldsymbol{u}^{(k)}, \boldsymbol{q}^{(k)})_{k=2}^{K}$. To this end, we also build the encoder-decoder architecture network whose parameters are partly shared with that at the first stage. Specifically, we employ the HS-Encoder same as the first stage, and design four decoders. Each layer in the decoder consists of a 1-D convolution, batch normalization, and dropout operation. We use the ReLU activation function for the final output of the layer. At the end of the network, we build four parallel branches for estimating joint parameters, joint states, per-part scales, and part labels respectively.

**Joint Parameter Branch.**   In order to recover per-part pose for the constrained part, it is required to estimate the joint parameters $(\boldsymbol{u}, \boldsymbol{q})$. Refer to A-NCSH [8] that predicts the joint parameters in normalized coordinate space, our EfficientCAPER predicts them in the articulation canonicalized space. To be specific, the joint parameter branch involves three heads: (1) joint direction head that regresses the joint direction $\boldsymbol{u}$ for both revolute and prismatic joints. (2) offset head that predicts the projected distance $o_i$ from each canonicalized point $\widetilde{p}_i$ to the joint $(\boldsymbol{u}, \boldsymbol{q})$. (3) heatmap head that predicts the score $h_i$ of the possibility of $p_i$ voting into the joint. In this way, the joint parameters $(\boldsymbol{u}, \boldsymbol{q})$ can be obtained by a voting scheme:

$$\boldsymbol{u} = \frac{1}{N}\sum_{i}^{N}\boldsymbol{u}_i \tag{2}$$

$$\boldsymbol{q} = \frac{1}{N}\sum_{i}^{N}h_i(\boldsymbol{q}_i + o_i) \tag{3}$$

Note that when estimating the parameters for prismatic joint, we only use joint direction head since $\boldsymbol{q}$ is not defined in prismatic motion.

**Joint State Branch.**   For estimating the poses of the constrained parts, we build a prediction branch that predicts the joint state $\boldsymbol{\theta}^{(k)}$ as pose replacement. The joint state branch ends with a 1-D convolution layer that outputs the final result containing $K-1$ channels corresponding to $K-1$ constrained parts in an articulated object. During inference for revolute joint, given the predicted joint parameters $\boldsymbol{\phi}_r^{(k)} = (\boldsymbol{u}_r^{(k)}, \boldsymbol{q}_r^{(k)})$ and joint states $\boldsymbol{\theta}_r^{(k)}$, we utilize the extended Rodrigues rotation formula to convert the joint state into a matrix $R_{\mathbf{cons}}^{(k)}$ for $k$-th part:

$$R_{\mathbf{cons}}^{'(k)} = \cos\boldsymbol{\theta}_r^{(k)}U_r^{(k)} + (1 - \cos\boldsymbol{\theta}_r^{(k)})(U_r^{(k)} \cdot Q_r^{(k)})Q_r^{(k)}) + \sin\boldsymbol{\theta}_r^{(k)}(Q_r^{(k)})^{\wedge} \tag{4}$$

where $Q_r^{(r)}$ represents a skew-symmetric matrix composed of pivot point position $\boldsymbol{q}_r^{(k)}$, $U_r^{(k)}$ represents the normalized direction of the joint axis $\boldsymbol{u}_r^{(k)}$. Here the $R_{\mathbf{cons}}^{'(k)}$ is a matrix of size $4 \times 3$ and we stack it with the row [0, 0, 0, 1] appended at the end to get relative homogeneous matrix $T_{\mathbf{cons}}^{'(k)}$ as $k$-th constrained part's pose.

In terms of prismatic joint, given joint parameters $(\boldsymbol{u}_p^{(k)})$ and predicted joint state $\boldsymbol{\theta}_p^{(k)}$, we can also get relative homogeneous matrix as pose for $k$-th part:

$$T_{\boldsymbol{cons}}^{'(k)} = \begin{bmatrix} \mathbf{I} & \boldsymbol{\theta}_p^{(k)} \boldsymbol{u}_p^{(k)} \\ 0 & 1 \end{bmatrix} \tag{5}$$

where $\mathbf{I}$ indicates the identity matrix. Finally, we can calculate the poses of constrained parts by $T_{\mathbf{cons}}^{(k)} = T_{\mathbf{free}} T_{\mathbf{cons}}^{'(k)}$.

**Part Scale Branch.** To predict the scale $\boldsymbol{s}^{(k)}$ for all the parts of an articulated object, we employ the regression scheme in the part scale branch that predicts the residual size $\boldsymbol{s}^{*(k)} = \boldsymbol{s}^{(k)} - \widetilde{\boldsymbol{s}}^{(k)}$ where $\widetilde{\boldsymbol{s}}^{(k)}$ is defined as the predefined mean size of $k$-th part. Therefore, the final scales are determined by $\boldsymbol{s}^{(k)} = \boldsymbol{s}^{*(k)} + \widetilde{\boldsymbol{s}}^{(k)}$.

## 4.6   Multi-Task Loss Function

We utilize multi-task loss functions to train our EfficientCAPER network. In the free part pose estimation stage, we first design the L1 loss function to supervise the $(r_m, r_n)$ vector and translation $t$ regression:

$$\mathcal{L}_{rot} = \|r_m - \hat{r}_m\|_1 + \|r_n - \hat{r}_n\|_1 , \mathcal{L}_{trans} = \left\|t - \hat{t}\right\|_1 \tag{6}$$

where $m$,$n$ represent the possible selected directions of normals. To train the confidence score $c$ that is used for $(r_m, r_n)$ voting, the loss function $\mathcal{L}_{conf}$ is defined as:

$$\mathcal{L}_{conf} = \sum_{\substack{i \in \{m,n\} \\ j \in \{1,2\}}} \left\|c_j - exp(-k_1|r_i - \hat{r}_i|^2)\right\|_1 \tag{7}$$

where $k_1$ is a factor $= -13.7$ which follows the same setting in GPV-Pose[24]. Regarding the constrained part pose estimation stage, we utilize four loss functions to supervise the joint parameter, joint state, part scale, and part segmentation branches. Firstly for joint parameter branch, the loss function $\mathcal{L}_{para}$ aims to optimize the projection distance from predicted joint location $\hat{\boldsymbol{q}}^{(k)}$ to joint $(\boldsymbol{u}^{(k)}, \boldsymbol{q}^{(k)})$:

$$\mathcal{L}_{loc} = \frac{1}{K-1} \sum_{k=2}^{K} Proj(\hat{\boldsymbol{q}}^{(k)}, \boldsymbol{u}^{(k)}, \boldsymbol{q}^{(k)}) \tag{8}$$

and we use the L1 loss to supervise the joint direction $\mathcal{L}_{dir}$. In terms of the joint state branch, since the joint states in the annotation can be positive or negative, directly regressing these joint states is quite difficult. So we design the joint state loss function $\mathcal{L}_{stat}$ with a positive penalty factor $\alpha = 20$:

$$\mathcal{L}_{stat} = \begin{cases} \frac{1}{K-1} \sum\limits_{k=2}^{K} \|\boldsymbol{\theta}^{(k)} - \hat{\boldsymbol{\theta}}^{(k)}\|_1, & if\,\boldsymbol{\theta}^{(k)}\hat{\boldsymbol{\theta}}^{(k)} > 0 \\ \frac{1}{K-1} \sum\limits_{k=2}^{K} \alpha\|\boldsymbol{\theta}^{(k)} - \hat{\boldsymbol{\theta}}^{(k)}\|_1, & if\,\boldsymbol{\theta}^{(k)}\hat{\boldsymbol{\theta}}^{(k)} < 0 \end{cases} \tag{9}$$

In this loss function, when the joint state of the $k$-th joint is opposite to the ground truth in annotation, we apply a penalty factor $\alpha$ to the loss of that particular joint which encourages the neural network to learn more correctly. Next, we employ L1 loss for part scale branch $\mathcal{L}_{sca}$ and Cross-Entropy loss for part segmentation branch $\mathcal{L}_{seg}$. Finally, the total training losses $\mathcal{L}_1$ for stage one and $\mathcal{L}_2$ for stage

two are both the weighted sum of the losses from free part pose estimation and constrained part pose estimation modules:

$$\mathcal{L}_1 = \lambda_{rot}\mathcal{L}_{rot} + \lambda_{trans}\mathcal{L}_{trans} + \lambda_{conf}\mathcal{L}_{conf} \tag{10}$$

$$\mathcal{L}_2 = \lambda_{loc}\mathcal{L}_{loc} + \lambda_{dir}\mathcal{L}_{dir} + \lambda_{stat}\mathcal{L}_{stat} + \lambda_{sca}\mathcal{L}_{sca} + \lambda_{seg}\mathcal{L}_{seg} \tag{11}$$

where the weight multiplication factors $\lambda_{rot}, \lambda_{trans}, \lambda_{conf}, \lambda_{loc}, \lambda_{dir}, \lambda_{stat}, \lambda_{sca}, \lambda_{seg}$ are set to be 8.0, 10.0, 1.0, 1.0, 1.0, 6.0, 1.0, 0.1. The optimal hyper-parameters are mined by cross-validation method.

# 5 Experiments

## 5.1 Experimental Settings

**Datasets.** We evaluate EfficientCAPER on three datasets: ArtImage [14], ReArtMix [13], and RobotArm [13]. First, ArtImage is a synthetic articulated object estimating dataset with the objects from PartNetMobility. This dataset contains five categories of articulated objects: Laptop, Eyeglasses, Dishwasher, Scissors, and Drawer. Next, our method has also been evaluated on a semi-synthetic dataset ReArtMix for articulated object estimating tasks. To validate the generality of our method in real-world scenarios, we finally train and test on the 7-part RobotArm dataset.

**Metrics and Implementation Details.** To validate the performance of the EfficientCAPER, we adopt degree error(°) for 3D rotation, distance error(m) for 3D translation, 3D IOU(%) for scale, and per image inference time for estimating speed. During the data pre-processing, input point clouds are sampled into 1,024 points as the network inputs. We have adopted the Ranger optimizer and the "flat and anneal" learning rate strategy. The learning rate remains $10^{-4}$ in stage "flat". The total training epoch is 250 for stage one and 200 for stage two. We put two trained models in the same script for inference to directly obtain per-part pose. All the experiments are implemented on an NVIDIA GeForce RTX 3090 GPU with 24GB memory.

Table 1: Comparison with state-of-the-arts on the ArtImage dataset. We validate our EfficientCAPER on categories Laptop, Eyeglasses, Dishwasher, Scissors and Drawer that contains 2, 3, 2, 2, 4 parts respectively. ↓ means the lower the better and ↑ means the upper the better.

| Category | Method | Per-part Pose | | | Inference Time per Image (s) ↓ |
| | | Rotation Error (°) ↓ | Translation Error (m) ↓ | 3D IOU (%) ↑ | |
|---|---|---|---|---|---|
| Laptop | A-NCSH [8] | 5.3, 5.4 | 0.054, **0.043** | 56.7, 40.2 | 9.0 |
| | OMAD [14] | 5.4, 4.3 | 0.062, 0.061 | 43.5, 24.1 | 1.6 |
| | ArtPERL [11] | 4.9, 4.7 | **0.053**, 0.066 | 64.6, 50.4 | 0.9 |
| | **EfficientCAPER** | **3.5, 4.0** | 0.065, 0.071 | **88.2, 87.9** | **0.02** |
| Eyeglasses | A-NCSH [8] | 3.7, 22.3, 23.2 | 0.049, 0.313, 0.324 | 52.5, 40.2, 39.6 | 11.9 |
| | OMAD [14] | 4.9, 7.5, 7.5 | 0.062, 0.103, 0.324 | 22.8, 20.5, 21.4 | 2.5 |
| | ArtPERL [11] | 4.1, **6.2**, 6.0 | 0.047, **0.095**, 0.091 | 58.6, 46.5, 51.7 | 1.0 |
| | **EfficientCAPER** | **3.1**, 7.2, **5.8** | **0.038**, 0.109, **0.089** | **91.8, 82.3, 84.3** | **0.02** |
| Dishwasher | A-NCSH [8] | 4.0, 4.8 | 0.059, 0.123 | 84.3, 56.2 | 5.5 |
| | OMAD [14] | 6.0, 6.2 | 0.104, 0.142 | 66.5, 38.9 | 1.6 |
| | ArtPERL [11] | 3.9, 4.3 | 0.055, **0.079** | 89.3, 67.6 | 0.9 |
| | **EfficientCAPER** | **2.5, 3.1** | **0.052**, 0.085 | **91.2, 83.9** | **0.02** |
| Scissors | A-NCSH [8] | **2.0**, 2.9 | 0.035, **0.025** | 46.5, 44.8 | 6.5 |
| | OMAD [14] | 3.9, 3.4 | 0.048, 0.039 | 35.6, 34.5 | 1.7 |
| | ArtPERL [11] | 2.2, **2.6** | **0.031**, 0.042 | 40.9, 46.3 | 0.8 |
| | **EfficientCAPER** | 5.0, 5.1 | 0.048, 0.099 | **74.2, 69.2** | **0.02** |
| Drawer | A-NCSH [8] | 2.8, 3.5, 3.9, 2.9 | 0.045, 0.155, 0.157, **0.075** | 90.2, 81.5, 78.4, 82.7 | 16.5 |
| | OMAD [14] | 4.4, 4.4, 4.4, 4.4 | 0.111, 0.143, 0.144, 0.115 | 75.8, 73.4, 70.2, 71.3 | 1.9 |
| | ArtPERL [11] | 3.5, 3.5, 3.5, 3.5 | 0.061, 0.112, 0.121, 0.104 | 84.8, 78.6, 79.0, 81.2 | 1.1 |
| | **EfficientCAPER** | **2.0, 2.0, 2.0, 2.0** | **0.041, 0.086, 0.090**, 0.080 | **93.2, 87.1, 87.1, 88.4** | **0.02** |

## 5.2 Experiments on ArtImage Dataset

We report the results of EfficientCAPER evaluated on the synthetic dataset containing the articulated objects from PartNet-Mobility in Table 1. As it can be seen, EfficientCAPER achieves optimal or suboptimal results on rotation error in most categories. We can see that the high estimating performance with only 2.0°, 2.6° on rotation error for category Dishwasher. This can be explained by that our joint-centric representation for articulation indicates a significant weakening of the impact of self-occlusion. For category Drawer, our method achieves excellent performance with only 0.041m, 0.086m, 0.090m, and 0.080m on translation error. This can be attributed to the framework's accurate

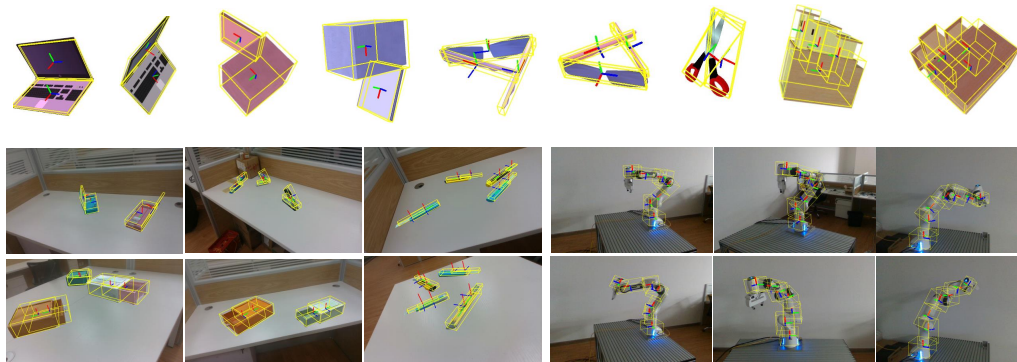

Figure 4: Qualitative results on ArtImage(top), ReArtMix(bottom left) and RobotArm(bottom right).

estimation for state and orientation of the prismatic joint. Obviously, our proposed method can predict 3D scale very well. In terms of inference time, many previous methods typically predict the NOCS coordinate for each observed point and rely on the Umeyama algorithm and the RANSAC method for pose recovery [12, 8], our method can regress the pose without any algorithmic optimization or retrieval method. EfficientCAPER achieves real-time prediction at a speed of approximately 50 FPS. Qualitative Results are shown in Figure 4.

## 5.3 Ablation Study

**Effect of Articulation Pose Canonicalization.** We investigate the effect of canonical point cloud for estimating joint states and 3D scale where we experiment on ArtImage and Table 2 illustrates the results. Note that we show joint state error and 3D scale score here since the articulation pose canonicalization module is only employed in the second stage of our EfficientCAPER. Here our 3D IoU calculation method is as follows: After predicting the length, width, and height of each part, we construct the predicted 3D bounding box and the ground truth 3D bounding box centered at [0,0,0]. We then calculate the 3D scale score between the two bounding boxes. It is obvious that canonicalization of point cloud proposed by us can improve estimating performance a lot since the network can be more sensitive to kinematic prior and shape prior in the canonical coordinate system.

Table 2: Experiments on the effect of articulation pose canonicalization (indicated as APC here) module.

| Category | APC | Joint State Error | 3D Scale Score(%) |
|---|---|---|---|
| Laptop | | 3. 5° | 83.8, 84.6 |
| | ✓ | **1.8°** | **86.2**, **86.8** |
| Eyeglasses | | 7.8°, 8.3° | 64.3, 54.8, 53.3 |
| | ✓ | **5.0°**, **3.7°** | **68.9**, **58.1**, **55.2** |
| Dishwasher | | 2.4° | 26.2, 59.6 |
| | ✓ | **1.8°** | **30.2**, **66.2** |
| Scissors | | 3.5° | **57.2**, **56.0** |
| | ✓ | **2.2°** | 56.5, 49.7 |
| Drawer | | 0.098m, 0.101m, 0.070m | 58.9, 57.5, 64.7, 65.2 |
| | ✓ | **0.075m**, **0.076m**, **0.050m** | **71.3**, **59.4**, **66.5**, **68.4** |

Table 3: Pose errors for *'Drawer'* category with varying occlusion levels.

| Occlusion Level | Rotation Error(°) | Translation Error(m) |
|---|---|---|
| 0%-40% | 1.9 | 0.057 |
| 40%-80% | 2.0 | 0.062 |
| 80%-100% | 2.0 | 0.095 |

**Self-Occlusion Analysis.** To further investigate the robustness under self-occlusion of our EfficientCAPER, we split the test samples of category Drawer into three subsets based on occlusion level since self-occlusion problems are more common in drawers. The occlusion level is defined as the visible ratio of the points compared to the whole part points, which is similar to A-NCSH [8]. In detail, there are three subsets with various visibility in this experiment: (0%, 40%), (40%, 80%) and (80%, 100%). Here we use mean rotation error and mean translation error that uniformly evaluate the pose estimation performance. Table 3 shows the experimental result. With an increasing occlusion level, rotation error remains essentially unchanged thanks to our joint-centric method. Our method achieves a small translation error for occlusion level rates up to 80%, even in extreme cases with occlusion rates ranging from 80% to 100%, our method still yields a favorable result with a translation error of 0.095m. Therefore, it can be concluded that our method works well in solving the self-occlusion problem. More ablation study can be found in Section A.5.1.

## 5.4 Generalization Capacity

**Experiments on Semi-Synthetic Scenarios.** First, we assess our method for the estimation of articulated object poses using the dataset known as ReArtMix, which incorporates semi-synthetic scenarios. The detailed results are shown in Table 4. Compared to the baseline, our method shows excellent performance on ReArtMix, obtaining better rotation estimation on all categories. We can see that the accurate translation estimation with only 0.003m, 0.005m on translation error for category Cutter. The qualitative results are shown in Figure 4.

**Experiments on Real-world Scenarios.** To investigate the performance in real-world scenarios, we also conduct training and evaluation of EfficientCAPER using the 7-part RobotArm dataset. It is important to acknowledge that the robot arm accumulates errors from the first part to the end effector due to the multi-depth structure of the robot arm instance, the results in Table 5 show that our method performs well with objects with diverse structures. More qualitative results are shown in Section A.6.

Table 4: Pose estimation results on ReArtMix dataset.

| Category | Method | Per-part Pose | | |
| --- | --- | --- | --- | --- |
| | | Rotation Error (°) | Translation Error (m) | 3D IOU(%) |
| Box | ReArtNet [13] | 3.4, 3.6 | 0.025, 0.034 | 52.1, 49.3 |
| | EfficientCAPER | **0.9**, **1.7** | **0.004**, **0.015** | **96.4**, **88.9** |
| Stapler | ReArtNet [13] | 4.9, 6.3 | 0.032, 0.030 | 40.1, 35.6 |
| | EfficientCAPER | **0.9**, **1.6** | **0.006**, **0.019** | **93.8**, **84.5** |
| Cutter | ReArtNet [13] | 3.0, 3.2 | 0.016, 0.017 | 35.5, 34.2 |
| | EfficientCAPER | **1.6**, **1.6** | **0.003**, **0.004** | **96.2**, **95.7** |
| Scissors | ReArtNet [13] | 5.6, 5.3 | 0.011, **0.013** | 26.3, 24.3 |
| | EfficientCAPER | **2.3**, **4.8** | **0.006**, 0.015 | **93.1**, **87.1** |
| Drawer | ReArtNet [13] | 3.1, 3.2 | 0.021, 0.019 | 51.5, 50.2 |
| | EfficientCAPER | **0.7**, **0.7** | **0.006**, **0.007** | **95.3**, **94.2** |

Table 5: Pose estimation results on 7-part RobotArm dataset

| Per-part Rotation Error (°) | | | | | | | |
| --- | --- | --- | --- | --- | --- | --- | --- |
| Part ID | 1 | 2 | 3 | 4 | 5 | 6 | 7 |
| A-NCSH [8] | 7.8 | 7.9 | 10.3 | 10.5 | 11.2 | 16.4 | 23.5 |
| EfficientCAPER | **0.04** | **5.9** | **7.6** | **7.3** | **7.7** | **11.6** | **14.6** |
| Per-part Translation Error (m) | | | | | | | |
| Part ID | 1 | 2 | 3 | 4 | 5 | 6 | 7 |
| A-NCSH [8] | **0.012** | 0.044 | 0.067 | 0.066 | **0.079** | 0.236 | 0.403 |
| EfficientCAPER | 0.014 | **0.020** | **0.048** | **0.049** | 0.126 | **0.124** | **0.304** |

## 6 Conclusion

In this paper, we propose an end-to-end approach, namely EfficientCAPER, for solving fast and robust category-level articulated object pose estimation problems. We formulate this task as a two-stage framework for the free part and constrained part pose estimation separately that utilizes joint-centric pose modeling to split a whole articulated object into two groups of parts. Experiments demonstrate that EfficientCAPER can obtain state-of-the-art estimating performance on various datasets and scenarios, with strong robustness in severe self-occlusion scenes. Future works will focus on investigating the articulated objects with various kinematic structures in the same category or cross-category setting.

## Acknowledgements

This research was supported by the Baima Lake Laboratory Joint Funds of the Zhejiang Provincial Natural Science Foundation of China under Grant No. LBMHD24F030002, the National Natural Science Foundation of China under Grant 62373329 and National Natural Science Foundation of China under Grant 62302143 and Anhui Provincial Natural Science Foundation under Grant 2308085QF207 and National Natural Science Foundation of China under Grant 62372150, and Beijing XiaoYu Intelligence Manufacturing Ltd Inc, Beijing China.

## Footnotes

*Corresponding author: Linlin Ou and Liu Liu.

# References

[1] Xinke Deng, Yu Xiang, Arsalan Mousavian, Clemens Eppner, Timothy Bretl, and Dieter Fox. Self-supervised 6d object pose estimation for robot manipulation. In *2020 IEEE International Conference on Robotics and Automation (ICRA)*, May 2020.

[2] Eric Marchand, Hideaki Uchiyama, and Fabien Spindler. Pose estimation for augmented reality: A hands-on survey. *IEEE Transactions on Visualization and Computer Graphics*, page 2633–2651, Dec 2016.

[3] Yongzhi Su, Jason Rambach, Nareg Minaskan, Paul Lesur, Alain Pagani, and Didier Stricker. Deep multi-state object pose estimation for augmented reality assembly. In *2019 IEEE International Symposium on Mixed and Augmented Reality Adjunct (ISMAR-Adjunct)*, Oct 2019.

[4] Yinyu Nie, Xiaoguang Han, Shihui Guo, Yujian Zheng, Jian Chang, and Jian Jun Zhang. Total3dunderstanding: Joint layout, object pose and mesh reconstruction for indoor scenes from a single image. In *2020 IEEE/CVF Conference on Computer Vision and Pattern Recognition (CVPR)*, Jun 2020.

[5] Yongming Wen, Yiquan Fang, Junhao Cai, Kimwa Tung, and Hui Cheng. Gccn: Geometric constraint co-attention network for 6d object pose estimation. In *Proceedings of the 29th ACM International Conference on Multimedia*, pages 2671–2679, 2021.

[6] Chen Wang, Danfei Xu, Yuke Zhu, Roberto Martín-Martín, Cewu Lu, Li Fei-Fei, and Silvio Savarese. Densefusion: 6d object pose estimation by iterative dense fusion. In *Proceedings of the IEEE/CVF conference on computer vision and pattern recognition*, pages 3343–3352, 2019.

[7] Yifei Shi, Junwen Huang, Xin Xu, Yifan Zhang, and Kai Xu. Stablepose: Learning 6d object poses from geometrically stable patches. In *Proceedings of the IEEE/CVF Conference on Computer Vision and Pattern Recognition*, pages 15222–15231, 2021.

[8] Xiaolong Li, He Wang, Li Yi, Leonidas J Guibas, A Lynn Abbott, and Shuran Song. Category-level articulated object pose estimation. In *Proceedings of the IEEE/CVF conference on computer vision and pattern recognition*, pages 3706–3715, 2020.

[9] Liu Liu, Wenqiang Xu, Haoyuan Fu, Sucheng Qian, Qiaojun Yu, Yang Han, and Cewu Lu. Akb-48: a real-world articulated object knowledge base. In *Proceedings of the IEEE/CVF Conference on Computer Vision and Pattern Recognition*, pages 14809–14818, 2022.

[10] Liu Liu, Han Xue, Wenqiang Xu, Haoyuan Fu, and Cewu Lu. Towards real-world category-level articulation pose estimation. *IEEE Transactions on Image Processing*, page 1072–1083, Jan 2022.

[11] Liu Liu, Jianming Du, Hao Wu, Xun Yang, Zhenguang Liu, Richang Hong, and Meng Wang. Category-level articulated object 9d pose estimation via reinforcement learning. In *Proceedings of the 31st ACM International Conference on Multimedia*, pages 728–736, 2023.

[12] He Wang, Srinath Sridhar, Jingwei Huang, Julien Valentin, Shuran Song, and Leonidas J. Guibas. Normalized object coordinate space for category-level 6d object pose and size estimation. In *2019 IEEE/CVF Conference on Computer Vision and Pattern Recognition (CVPR)*, Jun 2019.

[13] Liu Liu, Han Xue, Wenqiang Xu, Haoyuan Fu, and Cewu Lu. Toward real-world category-level articulation pose estimation. *IEEE Transactions on Image Processing*, 31:1072–1083, 2022.

[14] Han Xue, Liu Liu, Wenqiang Xu, Haoyuan Fu, and Cewu Lu. Omad: Object model with articulated deformations for pose estimation and retrieval. *arXiv preprint arXiv:2112.07334*, 2021.

[15] Wei Chen, Xi Jia, Hyung Jin Chang, Jinming Duan, Linlin Shen, and Ales Leonardis. Fs-net: Fast shape-based network for category-level 6d object pose estimation with decoupled rotation mechanism. In *Proceedings of the IEEE/CVF Conference on Computer Vision and Pattern Recognition*, pages 1581–1590, 2021.

[16] Linfang Zheng, Chen Wang, Yinghan Sun, Esha Dasgupta, Hua Chen, Aleš Leonardis, Wei Zhang, and Hyung Jin Chang. Hs-pose: Hybrid scope feature extraction for category-level object pose estimation. In *Proceedings of the IEEE/CVF Conference on Computer Vision and Pattern Recognition*, pages 17163–17173, 2023.

[17] Eric Brachmann, Alexander Krull, Frank Michel, Stefan Gumhold, Jamie Shotton, and Carsten Rother. Learning 6d object pose estimation using 3d object coordinates. In *Computer Vision– ECCV 2014: 13th European Conference, Zurich, Switzerland, September 6-12, 2014, Proceedings, Part II 13*, pages 536–551. Springer, 2014.

[18] Zhe Cao, Yaser Sheikh, and Natasha Kholgade Banerjee. Real-time scalable 6dof pose estimation for textureless objects. In *2016 IEEE International conference on Robotics and Automation (ICRA)*, pages 2441–2448. IEEE, 2016.

[19] Guangjun Shi, Xiangyang Xu, and Yaping Dai. Sift feature point matching based on improved ransac algorithm. In *2013 5th International Conference on Intelligent Human-Machine Systems and Cybernetics*, volume 1, pages 474–477. IEEE, 2013.

[20] Herbert Bay, Tinne Tuytelaars, and Luc Van Gool. Surf: Speeded up robust features. In *Computer Vision–ECCV 2006: 9th European Conference on Computer Vision, Graz, Austria, May 7-13, 2006. Proceedings, Part I 9*, pages 404–417. Springer, 2006.

[21] Bugra Tekin, Sudipta N Sinha, and Pascal Fua. Real-time seamless single shot 6d object pose prediction. In *Proceedings of the IEEE conference on computer vision and pattern recognition*, pages 292–301, 2018.

[22] Dengsheng Chen, Jun Li, Zheng Wang, and Kai Xu. Learning canonical shape space for category-level 6d object pose and size estimation. In *Proceedings of the IEEE/CVF conference on computer vision and pattern recognition*, pages 11973–11982, 2020.

[23] Haitao Lin, Zichang Liu, Chilam Cheang, Yanwei Fu, Guodong Guo, and Xiangyang Xue. Sar-net: Shape alignment and recovery network for category-level 6d object pose and size estimation. Jun 2021.

[24] Yan Di, Ruida Zhang, Zhiqiang Lou, Fabian Manhardt, Xiangyang Ji, Nassir Navab, and Federico Tombari. Gpv-pose: Category-level object pose estimation via geometry-guided point-wise voting. In *Proceedings of the IEEE/CVF Conference on Computer Vision and Pattern Recognition*, pages 6781–6791, 2022.

[25] Zhi-Hao Lin, Sheng-Yu Huang, and Yu-Chiang Frank Wang. Convolution in the cloud: Learning deformable kernels in 3d graph convolution networks for point cloud analysis. In *2020 IEEE/CVF Conference on Computer Vision and Pattern Recognition (CVPR)*, Jun 2020.

[26] Shih-En Wei, Varun Ramakrishna, Takeo Kanade, and Yaser Sheikh. Convolutional pose machines. In *Proceedings of the IEEE conference on Computer Vision and Pattern Recognition*, pages 4724–4732, 2016.

[27] Sijin Li and Antoni B Chan. 3d human pose estimation from monocular images with deep convolutional neural network. In *Computer Vision–ACCV 2014: 12th Asian Conference on Computer Vision, Singapore, Singapore, November 1-5, 2014, Revised Selected Papers, Part II 12*, pages 332–347. Springer, 2015.

# A    Appendix / supplemental material

## A.1    Overview

In the appendix, we present additional information and experiments of EfficientCAPER. An overview of our supplementary material goes as follows: Firstly, we discuss the limitations of the work in Section A.2. Then, we explain how we choose baseline and introduce more details of the network in Section A.3. In addition, we describe our training and inference strategies detailedly in Section A.4. To demonstrate the benefits of our approach, we have also provided some additional experimental results in Section A.5. Lastly, qualitative results on three datasets are shown in Section A.6.

## A.2    Limitations

Through analysis of the experimental results, our method heavily relies on free part pose estimation accuracy. Considering limited by the small size of the dataset and the relatively simple structures in current dataset, our method's superiority may not be fully realized. However, EfficientCAPER is robust to objects that have diverse structures such as robot-arm in the RobotArm dataset.

## A.3    The Choice of Baseline

### A.3.1    GPV-Pose

For free part pose estimation, we can treat it as a category-level rigid pose estimation task. We follow GPV-Pose [24] since its good utilization of geometric information, showing good performance and robustness to this task. Apart from rotation matrix decomposition and residual translation prediction, there are two other branches to supervise the pose: symmetric reconstruction and bounding box voting. All classes in ArtImage are symmetric about the $yoz$ plane, so we choose the y-axis and z-axis so that the symmetric reconstruction module can supervise the estimation of the plane normals in these two directions. Unlike GPV-Pose, we only supervise the rotation in bounding box voting. Apart from the aforementioned aspects, the design of the loss function and the selection of hyper-parameters for these two modules are consistent with GPV-Pose.

### A.3.2    HS-Pose

HS-Pose [16] proposed the HS-layer which extends from 3D-GC, can better perceive local and global geometric structure, encode translation and scale information. Due to the superiority of HS-layer, we have replaced 3D-GC with the HS-Encoder. We need to choose the essential hyper-parameters. The support number for the graph convolution network is 7 and the neighbor number for RF-F and ORL is 20.

## A.4    Training and Inference Protocol

**Training.**    In our approach, we train on each object category separately, and the entire training process is divided into two stages. In the first stage, we feed a partial point cloud to the HS-Econder to predict the pose of the free part. In the second stage, we need to utilize the pose of the free part to canonicalize the input point cloud, i.e. transforming the point cloud from the camera view to a "pseudo-canonical" view. "Pseudo" means the pose contains errors, the canonical space in this case is not standard. We have designed a special noise addition strategy to implement the training of the second stage. First, we can separately calculate the average rotation error and average translation error by evaluating the pose results of stage one. Then, we add noise to the gt pose to make the noisy pose have approximately the same rotation error and translation error compared to the ground truth. We use the noisy pose to canonicalize the point cloud and we feed the canonical point cloud to the other HS-Encoder for training the model of stage two.

**Inference.**    After training, we can separately obtain the best-performing models from stage one and stage two respectively. We integrate the two stages into a single inference script. First, we feed the point cloud to the model of stage one to get the pose of the free part. Then we use the pose to canonicalize the point cloud to get the canonical point cloud. In stage two, we input the canonical point cloud to the trained model, the model will output joint state estimation, joint parameters

estimation, 3D scale estimation, and part segmentation. Finally, the poses of the constrained parts are calculated based on the joint state, joint parameters, and the free part's pose. As discussed above, we can directly output the pose and 3D scale of every part.

## A.5 Experiments

### A.5.1 Ablation Study

**Single Free Part Input v.s. Whole Object Input.** We have conducted an additional set of experiments to validate the assistance of constrained part information in estimating the free part pose. Results are shown in Table 6. We can see that rotation error decreases when we use constrained parts because the neural network can learn better through a more comprehensive point cloud with a particularly significant increase of 90% in the category Dishwasher. However, we get a passable result on increased translation error since the original position of the free part may move after adding the constrained part. Overall, it is necessary to feed point clouds of all parts of articulation because it allows our two-stage network to simultaneously output the poses of all parts with just a single input.

Table 6: Experiments comparing single free part as input between both free part and constrained parts as input. Note that we only investigate this analysis in the first stage of our EfficientCAPER.

| Category | Input Parts | Rotation Error(°) | Translation Error(m) |
|---|---|---|---|
| Laptop | free | 3.8 | 0.059 |
| | free+constrained | 3.5 | 0.061 |
| Eyeglasses | free | 3.4 | 0.053 |
| | free+constrained | 3.1 | 0.037 |
| Dishwasher | free | 20.4 | 0.057 |
| | free+constrained | 2.5 | 0.053 |
| Scissors | free | 5.0 | 0.043 |
| | free+constrained | 5.0 | 0.048 |
| Drawer | free | 3.4 | 0.049 |
| | free+constrained | 2.0 | 0.043 |

### A.5.2 Detiled Results

**Joint Parameters Estimation Results.** In stage two, we also design a branch to estimate the parameters of the axes. All the joint parameters are evaluated in canonical space. For each revolute joint, we use mean angle error to evaluate the orientation of the joint axis, and use mean distance error to evaluate the distance between the predicted axis position and the gt axis position. For each prismatic joint, we compute the orientation error of the translation axis. The results are shown in Table 7, Table 8 and Table 9. For orientation, our EfficientCAPER can obtain good performance on all classes. For location, it is obvious that our method can still achieve satisfactory results on three datasets.

Table 7: Experiments estimating joint parameters in the canonical space on ArtImage.

| Category | Joint Parameter | |
|---|---|---|
| | Angle Error (°) | Distance Error (m) |
| Laptop | 0.00 | 0.037 |
| Eyeglasses | 0.00, 0.67 | 0.031, 0.024 |
| Dishwasher | 0.00 | 0.033 |
| Scissors | 0.00 | 0.097 |
| Drawer | 0.04, 0.09, 0.00 | -, -, - |

Table 8: Experiments estimating joint parameters in the canonical space on ReArtMix.

| Category | Joint Parameter | |
|---|---|---|
| | Angle Error (°) | Distance Error (m) |
| Box | 0.43 | 0.011 |
| Stapler | 0.63 | 0.012 |
| Cutter | 0.48 | - |
| Scissors | 0.80 | 0.005 |
| Drawer | 0.11 | - |

**Part Segmentation Results.** In stage two, our model can output part segmentation results which is an optional branch. We conduct two sets of experiments on part segmentation in both camera view and canonical view, and we calculate the mean IOU of all parts. The results shown in Table 10 demonstrate that our framework can achieve performance with an IOU greater than 90% on four categories of objects. It is proved that articulation pose canonicalization can also improve the part segmentation performance.

Table 9: Experiments estimating joint parameters in the canonical space on 7-part RobotArm dataset

| Joint Parameter | | | | | | |
|---|---|---|---|---|---|---|
| Joint ID | 1 | 2 | 3 | 4 | 5 | 6 |
| Angle Error (°) | 0.005 | 0.008 | 0.019 | 0.114 | 0.018 | 0.012 |
| Distance Error (m) | 0.012 | 0.013 | 0.014 | 0.028 | 0.023 | 0.038 |

Table 10: Part Segmentation results. APC means articulation pose canonicalization module.

| Category | APC | Mean IOU(%) |
|---|---|---|
| Laptop | | 94.3 |
| | ✓ | 95.0 |
| Eyeglasses | | 88.3 |
| | ✓ | 90.1 |
| Dishwasher | | 96.7 |
| | ✓ | 97.7 |
| Scissors | | 88.47 |
| | ✓ | 77.5 |
| Drawer | | 79.4 |
| | ✓ | 92.3 |

## A.6 Qualitative Results

In this section, we show more qualitative results on ArtImage [14], ReArtMix [13], and RobotArm [13] datasets. In Figure 5, we show the visualizations of Ground Truth, A-NCSH and our method on ArtImage. It can be observed that compared to A-NCSH, our method is capable of better predicting the 3D bounding boxes of each part of the object in camera space and gets very close results to the Ground Truth on Laptop, Dishwasher and Drawer. Figure 6 and Figure 7 are the results from ReArtMix and RobotArm. From these results, it can be seen that our method can accurately estimate the pose and 3D scale even in some severe self-occlusion views.

## A.7 Ethics Statement and Boarder Impact

Our method has the potential to develop the home-assisting robot, thus contributing to social welfare. We evaluate our method in synthesized or human-collected datasets, which may introduce data bias. However, similar studies also have such general concerns. We do not see any possible major harm in our study.

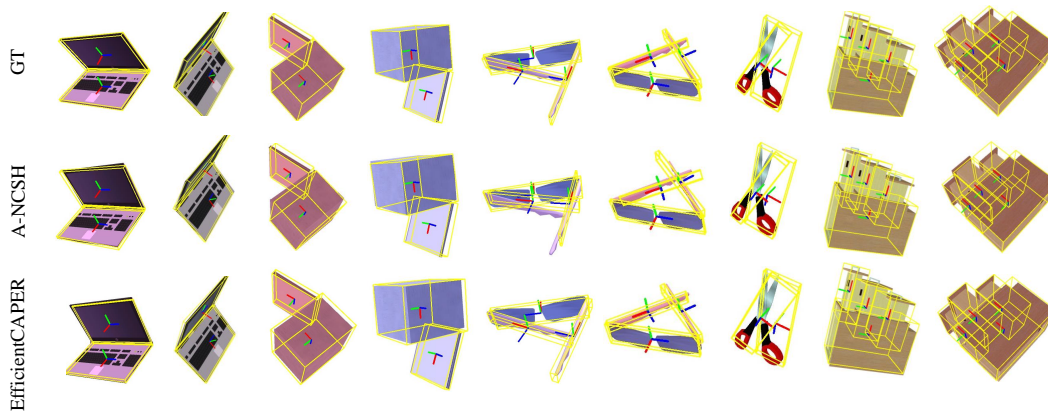

Figure 5: Qualitative results on the ArtImage dataset. We show five categories from ArtImage. The baseline is A-NCSH [8].

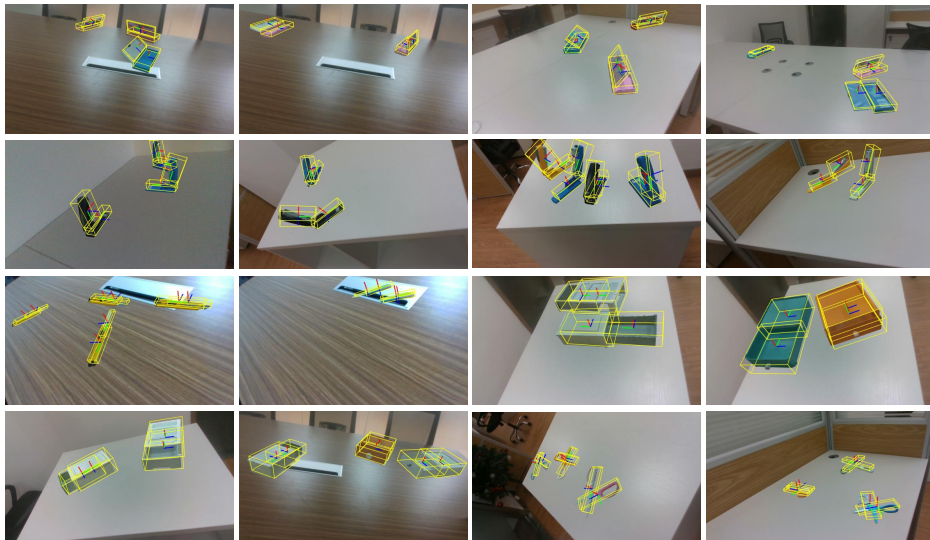

Figure 6: Qualitative results on the ReArtMix dataset.

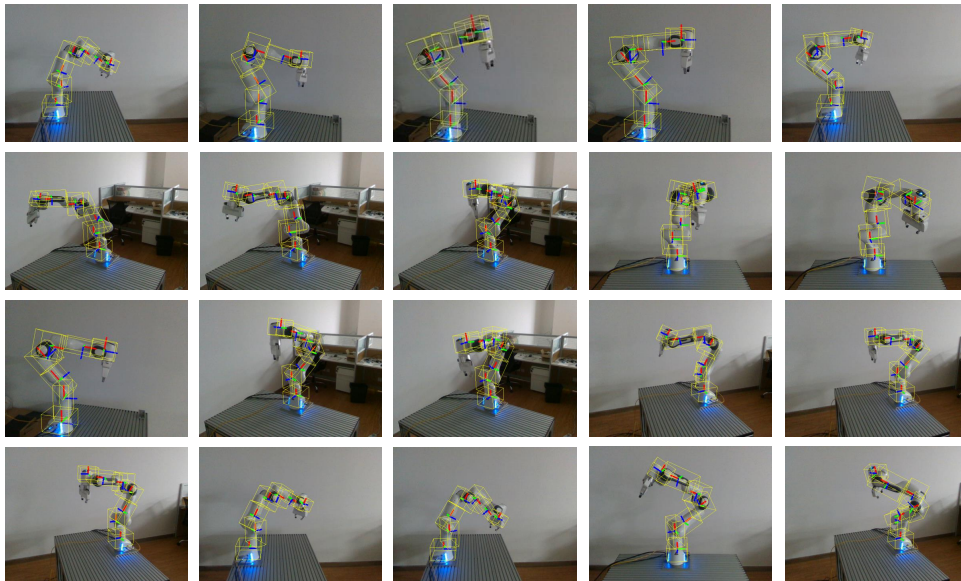

Figure 7: Qualitative results on the RobotArm dataset.

